# Hierarchical spike coding of sound

**Yan Karklin**[*]
Howard Hughes Medical Institute,
Center for Neural Science
New York University
yan.karklin@nyu.edu

**Chaitanya Ekanadham**[*]
Courant Institute of Mathematical Sciences
New York University
chaitu@math.nyu.edu

**Eero P. Simoncelli**
Howard Hughes Medical Institute, Center for Neural Science,
and Courant Institute of Mathematical Sciences
New York University
eero.simoncelli@nyu.edu

## Abstract

Natural sounds exhibit complex statistical regularities at multiple scales. Acoustic events underlying speech, for example, are characterized by precise temporal and frequency relationships, but they can also vary substantially according to the pitch, duration, and other high-level properties of speech production. Learning this structure from data while capturing the inherent variability is an important first step in building auditory processing systems, as well as understanding the mechanisms of auditory perception. Here we develop Hierarchical Spike Coding, a two-layer probabilistic generative model for complex acoustic structure. The first layer consists of a sparse spiking representation that encodes the sound using kernels positioned precisely in time and frequency. Patterns in the positions of first layer spikes are learned from the data: on a coarse scale, statistical regularities are encoded by a second-layer spiking representation, while fine-scale structure is captured by recurrent interactions within the first layer. When fit to speech data, the second layer acoustic features include harmonic stacks, sweeps, frequency modulations, and precise temporal onsets, which can be composed to represent complex acoustic events. Unlike spectrogram-based methods, the model gives a probability distribution over sound pressure waveforms. This allows us to use the second-layer representation to synthesize sounds directly, and to perform model-based denoising, on which we demonstrate a significant improvement over standard methods.

## 1 Introduction

Natural sounds, such as speech and animal vocalizations, consist of complex acoustic events occurring at multiple scales. Precise timing and frequency relationships among these events convey important information about the sound, while intrinsic variability confounds simple approaches to sound processing and understanding. Speech, for example, can be described as a sequence of words, which are composed of precisely interrelated phones, but each utterance may have its own prosody, with variable duration, loudness, and/or pitch. An auditory representation that captures the corresponding structure while remaining invariant to this variability would provide a useful first step for many applications in auditory processing.

[*]Contributed equally

Many recent efforts to learn auditory representations in an unsupervised setting have focused on sparse decompositions chosen to capture structure inherent in sound ensembles. The dictionaries can be chosen by hand [1, 2] or learned from data. For example, Klein et al [3] adapted a set of time-frequency kernels to represent spectrograms of speech signals and showed that the resulting kernels were localized and bore resemblance to auditory receptive fields. Lee et al [4] trained a two-layer deep belief network on spectrogram patches and used it for several auditory classification tasks.

These approaches have several limitations. First, they operate on spectrograms (rather than the original sound waveforms), which impose limitations on both time and frequency resolution. In addition, most models built on spectrograms rely on block-based partitioning of time, and thus are susceptible to artifacts – precisely-timed acoustic events can appear across multiple blocks and events can appear at different temporal offsets relative to the block, making their identification and representation difficult [5]. The features learned by these models are tied to specific frequencies, and must be replicated at different frequency offsets to accommodate pitch shifts that occur in natural sounds. Finally, the *linear* generative models underlying most methods are unsuitable for constructing hierarchical models, since the composition of multiple linear stages is again linear.

To address these limitations, we propose a two-layer hierarchical model that encodes complex acoustic events using a representation that is shiftable in both time and frequency. The first layer is a "spikegram" representation of the sound pressure waveform, as developed in [6, 5]. The prior probabilities for coefficients in the first layer are modulated by the output of the second layer, combined with a recurrent component that operates within the first layer. When trained on speech, the kernels learned at the second layer encode complex acoustic events which, when positioned at specific times and frequencies, compactly represent the first-layer spikegram, which is itself a compact description of the sound pressure waveform. Despite its very sparse activation, the second-layer representation retains much of the acoustic information: sounds sampled according to the generative model approximate well the original sound. Finally, we demonstrate that the model performs well on a denoising task, particularly when the noise is structured, suggesting that the higher-order representation provides a useful statistical description of speech.

## 2    Hierarchical spike coding

In the "spikegram" representation [5], a sound is encoded using a linear combination of sparse, time-shifted kernels $\phi_f(t)$:

$$x_t = \sum_{\tau,f} S_{\tau,f} \phi_f(t - \tau) + \epsilon_t \tag{1}$$

where $\epsilon_t$ denotes Gaussian white noise and the coefficients $S_{\tau,f}$ are mostly zero. As in [5], the $\phi_f(t)$ are gammatone functions with varying center frequencies, indexed by $f$. In order to encode the signal, a sparse set of "spikes" (i.e., nonzero coefficients at specific times and frequencies) is estimated using an approximate inference method, such as matching pursuit [7]. The resulting spikegram, shown in Fig. 1b, offers an efficient representation of sounds [8] that avoids the blocking artifacts and time-frequency trade-offs associated with more traditional spectrogram representations.

We aim to model the statistical regularities present in the spikegram representations. Spikegrams exhibit clear statistical structure, both at coarse (Fig. 1b,c) and at fine temporal scales (Fig. 1e,f). Spikes placed at precise locations in time and frequency reveal acoustic features, harmonic structures, as well as slow modulations in the sound envelope. The coarse scale non-stationarity is likely caused by higher-order acoustic events, such as phoneme utterances that span a much larger time-frequency range than the individual gammatone kernels. On the other hand, the fine-scale correlations are due to some combination of the correlations inherent in the gammatone filterbank and the precise temporal structure present in speech.

We introduce the hierarchical spike coding (HSC) model, illustrated in Fig. 2, to capture the structure in the spikegrams ($S^{(1)}$) on both coarse and fine scales. We add a second layer of unobserved spikes ($S^{(2)}$), assumed to arise from a Poisson process with constant rate $\lambda$. These spikes are convolved with a set of time-frequency "rate kernels" ($K^r$) to yield the logarithm of the firing rate of the first-layer spikes on a coarse scale. On a fine scale, the logarithm of the firing rate of first-layer spikes is modulated using recurrent interactions, by convolving the local spike history with

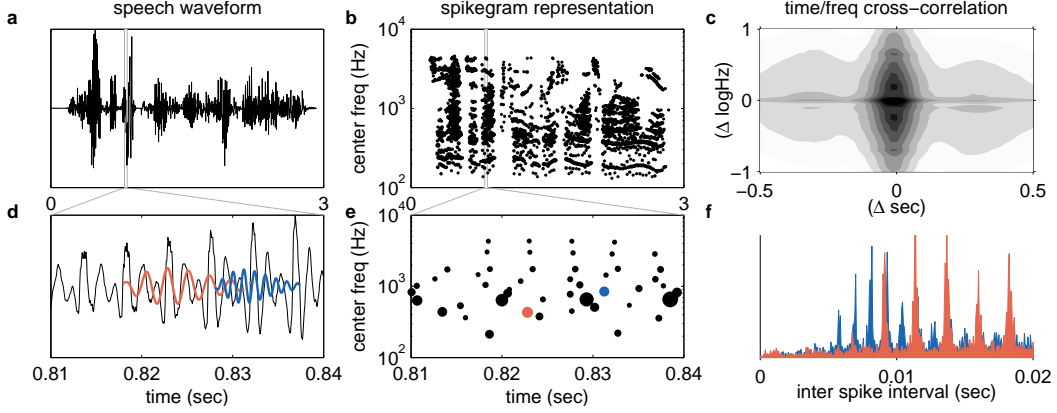

Figure 1: Coarse (top row) and fine (bottom row) scale structure in spikegram encodings of speech. **a**. The sound pressure waveform of a spoken sentence and **b**. the corresponding spikegram. Each spike (dot) has an associated time (abscissa) and center frequency (ordinate) as well as an amplitude (dot size). **c**. Cross-correlation function for a spikegram ensemble reveals correlations across large time/frequency scales. **d**. Magnification of a portion of (**a**), with two gammatone kernels (red and blue), corresponding to the red and blue spikes in (**e**). **e**. Magnification of corresponding portion of (**b**), revealing that spike timing exhibits strong regularities at a fine scale. **f**. Histograms of inter-spike-intervals for two frequency channels corresponding to the colored spikes in (**e**) reveal strong temporal dependencies.

a set of "coupling kernels" ($K^c$). The amplitudes of the first-layer spikes are also specified hierarchically: the logarithm of the amplitudes is assumed to be normally distributed, with a mean specified by the convolution of second-layer spikes with "amplitude kernels", ($K^a$ not shown) without any recurrent contribution, and the variance fixed at $\sigma^2$. The model parameters are denoted by $\Theta = \left( K^r, K^a, K^c, \vec{b}^r, \vec{b}^a \right)$ where $\vec{b}^r, \vec{b}^a$ are the bias vectors corresponding to the log-rate and log-amplitude of the first-layer coefficients, respectively. The model specifies a conditional probability density over first-layer coefficients,

$$P(S_{t,f}^{(1)}|S^{(2)};\Theta) = (1-p)\,\delta(S_{t,f}^{(1)}) + p\mathcal{N}\left(\log S_{t,f}^{(1)}; A_{t,f}, \sigma^2\right) \qquad \text{for } S_{t,f}^{(1)} \geq 0,\ \forall t, f \qquad (2)$$

$$\text{where } p = \Delta_t \Delta_f e^{R_{t,f}} \qquad \text{and} \qquad \mathcal{N}\left(x; \mu, \sigma^2\right) = \frac{e^{-\frac{(x-\mu)^2}{2\sigma^2}}}{\sqrt{2\pi\sigma^2}} \qquad (3)$$

$$R_{t,f} = b_f^r + (K^c * \mathbf{1}_{S^{(1)}})_{t,f} + \sum_i \left[ (K_i^r * S_i^{(2)})_{t,f} \right] \qquad (4)$$

$$A_{t,f} = b_f^a + \sum_i \left[ (K_i^a * S_i^{(2)})_{t,f} \right] \qquad (5)$$

In Eq. (2), $\delta(.)$ is the Dirac delta function. In Eq. (3), $\Delta_t$ and $\Delta_f$ are the time and frequency bin sizes. In Eqs. (4-5), $*$ denotes convolution and $\mathbf{1}_x$ is 1 if $x \neq 0$, and 0 otherwise.

## 3 Learning

The joint log-probability of the first and second layer can be expressed as a function of the model parameters $\Theta$ and the (unobserved) second-layer spikes $S^{(2)}$:

$$\mathcal{L}(\Theta, S^{(2)}) = \log P(S^{(1)}, S^{(2)}; \Theta, \lambda) = \log P(S^{(1)}|S^{(2)};\Theta) + \log P(S^{(2)};\lambda) \qquad (6)$$

$$= \sum_{(t,f)\in S^{(1)}} \left( R_{t,f} - \frac{1}{2\sigma^2}\left(\log S_{t,f}^{(1)} - A_{t,f}\right)^2 \right) - \sum_{t,f} e^{R_{t,f}}\Delta_t\Delta_f \qquad (7)$$

$$- \log\left(\lambda\Delta_t\Delta_f\right)\|S^{(2)}\|_0 + \text{const}$$

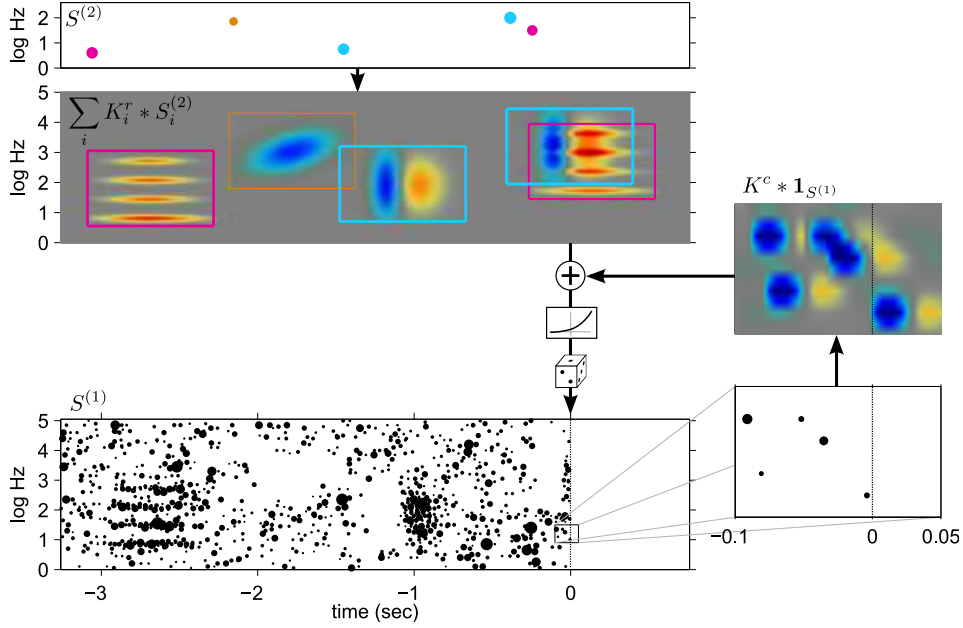

Figure 2: Illustration of the hierarchical spike coding model. Second-layer spikes $S^{(2)}$ associated with 3 features (indicated by color) are sampled in time and frequency according to a Poisson process, with exponentially-distributed amplitudes (indicated by dot size). These are convolved with corresponding rate kernels $K^r$ (outlined in colored rectangles), summed together, and passed through an exponential nonlinearity to drive the instantaneous rate of the first-layer spikes on a coarse scale. The first-layer spike rate is also modulated on a fine scale by a recurrent component that convolves previous spikes with coupling kernels $K^c$. At a given time step (vertical line), spikes $S^{(1)}$ are generated according to a Poisson process whose rate depends on the top-down and the recurrent terms.

where the equality in Eq. (7) holds in the limit $\Delta_t \Delta_f \to 0$. Maximizing the data likelihood requires integrating $\mathcal{L}$ over all possible second-layer representations $S^{(2)}$, which is computationally intractable. Instead, we choose to approximate the optimal $\Theta$ by maximizing $\mathcal{L}$ jointly over $\Theta$ and $S^{(2)}$. If $S^{(2)}$ is known, then the model falls within the well-known class of generalized linear models (GLMs) [9], and Eq. (6) is convex in $\Theta$. Conversely, if $\Theta$ is known then Eq. (6) is convex in $S^{(2)}$ except for the $L_0$ penalty term corresponding to the prior on $S^{(2)}$. Motivated by these facts, we adopt a coordinate-descent approach by alternating between the following steps:

$$S^{(2)} \leftarrow \arg\max_{S^{(2)}} \mathcal{L}(\Theta, S^{(2)}) \tag{8}$$

$$\Theta \leftarrow \Theta + \eta \nabla_\Theta \mathcal{L}(\Theta, S^{(2)}) \tag{9}$$

where $\eta$ is a fixed learning rate. Section 4 describes a method for approximate inference of the second-layer spikes (solving Eq. (8)). The gradients used in Eq. (9) are straightforward to compute and are given by

$$\frac{\partial \mathcal{L}}{\partial b_f^r} = (\# \, 1' \text{ spikes in channel } f) - \sum_t e^{R_{t,f}} \Delta_t \Delta_f \tag{10}$$

$$\frac{\partial \mathcal{L}}{\partial b_f^a} = \frac{1}{\sigma^2} \sum_t \left( \log S_{t,f}^{(1)} - A_{t,f} \right) \tag{11}$$

$$\frac{\partial \mathcal{L}}{\partial K_{\tau,\zeta,i}^r} = \sum_{(t,f) \in S^{(1)}} S_i^{(2)}(t-\tau, f-\zeta) - \sum_{t,f} e^{R_{t,f}} S_{t-\tau,f-\zeta,i}^{(2)} \Delta_t \Delta_f \tag{12}$$

$$\frac{\partial \mathcal{L}}{\partial K_{\tau,f,f'}^c} = \sum_{t \in S_f^{(1)}} \mathbf{1}_{S_{t-\tau,f'}^{(1)}} - \sum_t e^{R_{t,f}} \mathbf{1}_{S_{t-\tau,f'}^{(1)}} \Delta_t \Delta_f \tag{13}$$

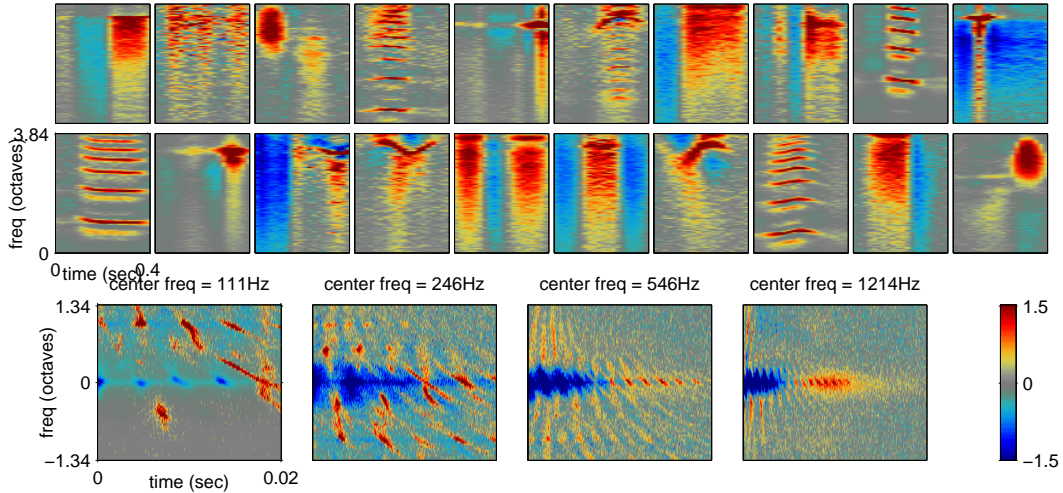

Figure 3: Example model kernels learned on the TIMIT data set. Top: rate kernels (colormaps individually rescaled). Bottom: Four representative coupling kernels (scaling indicated by colorbar).

## 4 Inference

Inference of the second-layer spikes $S^{(2)}$ (Eq. (8)) involves maximizing the trade-off between the GLM likelihood term, which we denote by $\tilde{\mathcal{L}}(\Theta, S^{(2)})$ and the last term which penalizes the number of spikes ($\|S^{(2)}\|_0$). Solving Eq. (8) exactly is NP-hard. We adopt a variant of the well-known matching pursuit algorithm [7] to approximate the solution. First, $S^{(2)}$ is initialized to $\vec{0}$. Then the following two steps are repeated:

1. Select the coefficient that maximizes a second-order Taylor approximation of $\tilde{\mathcal{L}}(\Theta, \cdot)$ about the current solution $S^{(2)}$:

$$(\tau^*, \zeta^*, i^*) = \arg\max_{\tau, \zeta, i} - \left( \frac{\partial \tilde{\mathcal{L}}}{\partial S^{(2)}_{\tau, \zeta, i}} \right)^2 \Big/ \frac{\partial^2 \tilde{\mathcal{L}}}{\partial S^{(2)\,2}_{\tau, \zeta, i}} \tag{14}$$

2. Perform a line search to determine the step size for this coefficient that maximizes $\tilde{\mathcal{L}}(\Theta, \cdot)$. If the maximal improvement does not outweigh the cost $-\log(\lambda \Delta_t \Delta_f)$ of adding a spike, terminate. Otherwise update $S^{(2)}$ using this step and repeat Step 1.

## 5 Results

**Model parameters learned from speech**

We applied the model to the TIMIT speech corpus [10]. First, we obtained spikegrams by encoding sounds to 20dB precision using a set of 200 gammatone filters with center frequencies spaced evenly on a logarithmic scale (see [5] for details). For each audio sample, this gave us a spikegram with fine time and frequency resolution ($6.25 \times 10^{-5}$s and $3.8 \times 10^{-2}$ octaves, respectively). We trained a model with 20 rate and 20 amplitude kernels, with frequency resolution equivalent to that of the spikegram and time resolution of 20ms. These kernels extended over 400ms×3.8 octaves (spanning 20 time and 100 frequency bins). Coupling kernels were defined independently for each frequency channel; they extended over 20ms and 2.7 octaves around the channel center frequency with the same time/frequency resolution as the spikegram. All parameters were initialized randomly, and were learned according to Eq. (8-9).

Fig. 3 displays the learned rate kernels (top) and coupling kernels (bottom). Among the patterns learned by the rate kernels are harmonic stacks of different durations and pitch shifts (e.g., kernels 4, 9, 11, 18), ramps in frequency (kernels 1, 7, 15, 16), sharp temporal onsets and offsets (kernels

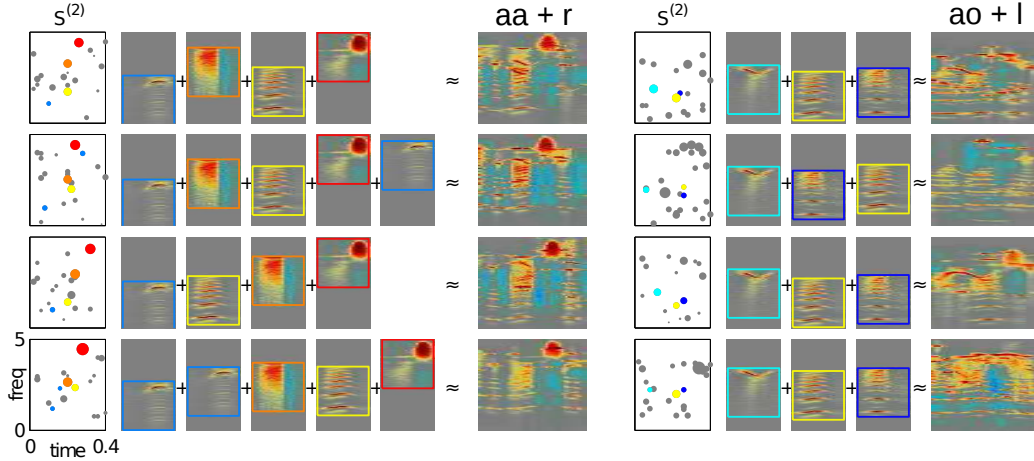

Figure 4: Model representation of phone pairs *aa+r* (left) and *ao+l* (right), as uttered by four speakers (rows: two male, two female). Each row shows inferred second-layer spikes, the rate kernels most correlated with the utterance of each phone pair, shifted to their corresponding spikes' frequencies (colored on left), and the encoded log firing rate centered on the phone pair utterance.

7, 13, 19), and acoustic features localized in time and frequency (kernels 5, 10, 12, 20) (example sounds synthesized by turning on single features are available in supplementary materials). The corresponding amplitude kernels (not shown) contain patterns highly correlated with the rate kernels, suggesting a strong dependence in the spikegram between spike rate and magnitude. For most frequency channels, the coupling kernels are strongly negative at times immediately following the spike and at adjacent frequencies, representing "refractory periods" observed in the spikegrams. Positive peaks in the coupling kernels encode precise alignment of spikes across time and frequency.

**Second-layer representation**

The learned kernels combine in various ways to represent complex acoustic events. For example, Fig. 4 illustrates how features can combine to represent two different phone pairs. Vowel phones are approximated by a harmonic stack (outlined in yellow) together with a ramp in frequency (outlined in orange and dark blue). Because the rate kernels add to specify the logarithm of the firing rate, their superposition results in a multiplicative modulation of the intensities at each level of the harmonic stack. In addition, the 'r' consonant in the first example is characterized by a high concentration of energy at the high frequencies and is largely accounted for by the kernel outlined in red. The 'l' consonant following 'ao' contains a frequency modulation captured by the v-shaped feature (outlined in cyan).

Translating the kernels in log-frequency allows the same set of fundamental features to participate in a range of acoustic events: the same vocalizations at different pitch are often represented by the same set of features. In Fig. 4, the same set of kernels is used in a similar configuration across different speakers and genders. It should be noted that the second-layer representation does not discard precise time and frequency information (this information is carried in the times and frequencies of the second-layer spikes). However, the identities of the features that are active remain invariant to pitch and frequency modulations.

**Synthesis**

One can further understand the acoustic information that is captured by second-layer spikes by sampling a spikegram according to the generative model. We took the second-layer encoding of a single sentence from the TIMIT speech corpus [10] (Fig. 5 middle) and sampled two spikegrams: one with only the hierarchical component (left), and one that included both hierarchical and coupling components (right). At a coarse scale the two samples closely resemble the spikegram of the original sound. However, at the fine time scale, only the spikegram sampled with coupling contains the regularities observed in speech data (Fig. 5 bottom row). Sounds were also generated from these spikegram samples by superimposing gammatone kernels as in [5]. Despite the fact that the second-

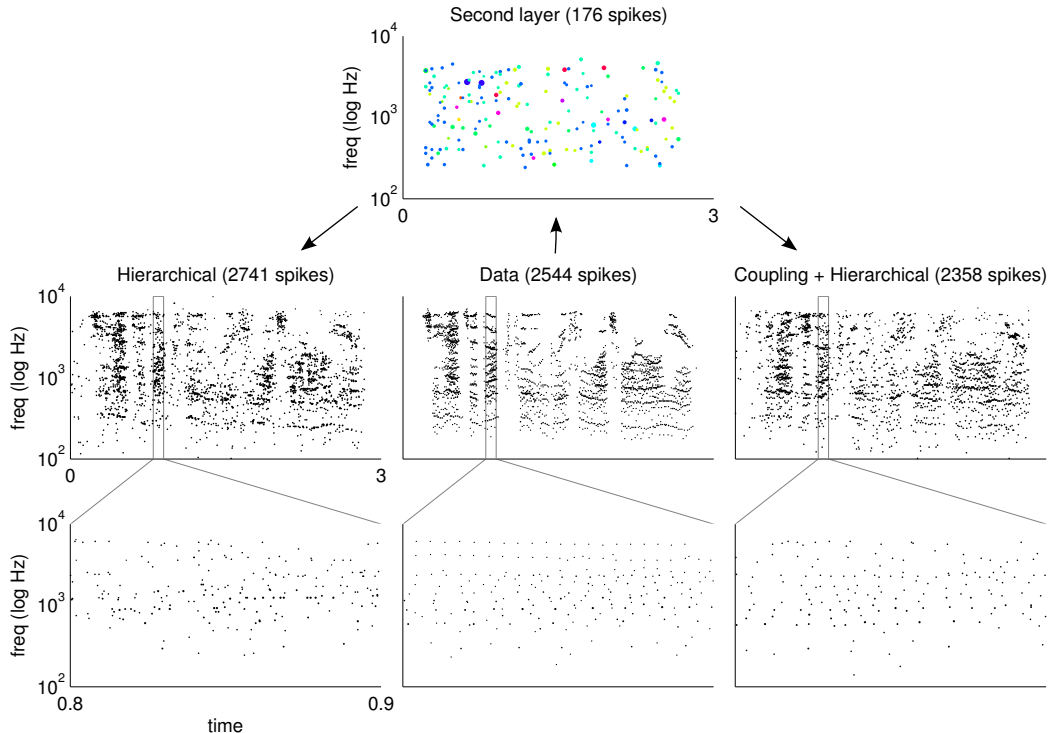

Figure 5: Synthesis from inferred second-layer spikes. Middle bottom: spikegram representation of the sentence in Fig. 1; Middle top: Inferred second-layer representation; Left: first-layer spikes generated using only the hierarchical model component; Right: first-layer spikes generated using hierarchical and coupling kernels. Synthesized waveforms are included in the supplementary materials.

| | white noise | | | | | sparse temporally modulated noise | | | |
|---|---|---|---|---|---|---|---|---|---|
| noise level | Wiener | wav thr | MP | HSC | | Wiener | wav thr | MP | HSC |
| -10dB | -7.00 | 2.41 | 2.26 | **2.50** | -10dB | -8.68 | -8.73 | -5.12 | **-4.37** |
| -5dB | 0.00 | 4.93 | 4.79 | **5.01** | -5dB | -3.09 | -3.63 | -0.96 | **-0.38** |
| 0dB | 5.49 | 7.94 | 7.71 | **7.99** | 0dB | 1.90 | 1.23 | 2.97 | **3.30** |
| 5dB | 7.84 | 11.15 | 11.01 | **11.33** | 5dB | 6.37 | 6.06 | 7.11 | **7.40** |
| 10dB | 10.31 | 14.64 | 14.49 | **14.83** | 10dB | 9.68 | 11.28 | 11.58 | **11.88** |

Table 1: Denoising accuracy (dB SNR) for speech corrupted with white noise (left) or with sparse, temporally modulated noise (right).

layer representation contains over 15 times fewer spikes as the first-layer spikegrams, the synthesized sounds are intelligible and the addition of the coupling filters provides a noticeable improvement (audio examples in supplementary materials).

**Denoising**

Although the model parameters have been adapted to the data ensemble, obtaining an estimate of the likelihood of the data ensemble under the model is difficult, as it requires integrating over unobserved variables ($S^{(2)}$). Instead, we can use performance on unsupervised signal processing tasks, such as denoising, to validate the model and compare it to other methods that explicitly or implicitly represent data density. In the noiseless case, a spikegram is obtained by running matching pursuit until the decrease in the residual falls below a threshold; in the presence of noise, this encoding process can be formulated as a denoising operation, terminated when the improvement in the log-likelihood (variance of the residual divided by the variance of the noise) is less than the cost of adding a spike (the negative log-probability of spiking). We incorporate the HSC model directly into this denoising algorithm by replacing the fixed probability of spiking at the first layer with the

rate specified by the second layer. Since neither the first- nor second-layer spike code for the noisy signal is known, we first infer the first and then the second layer using MAP estimation, and then recompute the first layer given both the data and second layer. The denoised waveform is obtained by reconstructing from the resulting first-layer spikes.

To the extent that the parameters learned by HSC reflect statistical properties of the signal, incorporating the more sophisticated spikegram prior into a denoising algorithm should allow us to better distinguish signal from noise. We tested this by denoising speech waveforms (held out during model training) that have been corrupted by additive white Gaussian noise. We compared the model's performance to that of the matching pursuit encoding (sparse signal representation without a hierarchical model), as well as to two standard denoising methods, Wiener filtering and wavelet-threshold denoising (implemented with MATLAB's wden function, using symlets, SURE estimator for soft threshold selection; other parameters optimized for performance on the training data set) [11].

HSC-based denoising is able to outperform standard methods, as well as matching pursuit denoising (Table 1 left). Although the performance gains are modest, the fact that the HSC model, which is not optimized for the task or trained on noisy data, can match the performance of adaptive algorithms like wavelet filtering denoising suggests that it has learned a representation that successfully exploits the statistical regularities present in the data.

To test more rigorously the benefit of a structured prior, we evaluated denoising performance on signals corrupted with non-stationary noise whose power is correlated over time. This is a more challenging task, but it is also more relevant to real-world applications, where sources of noise are often non-stationary. Algorithms that incorporate specific (but often incorrect) noise models (e.g., Wiener filtering) tend to perform poorly in this setting. We generated sparse temporally modulated noise by scaling white Gaussian noise with a temporally smooth envelope (given as a convolution of a Gaussian function with st. dev. of 0.02s with a Poisson process with rate $16s^{-1}$). All methods fare worse on this task. Again, the hierarchical model outperforms other methods (Table 1 right), but here the improvement in performance is larger, especially at high noise levels where the model prior plays a greater role. The reconstruction SNR does not fully convey the manner in which different algorithms handle noise: perceptually, we find that the sounds denoised by the hierarchical model sound more similar to the original (audio examples in supplementary materials).

## 6 Discussion

We developed a hierarchical spike code model that captures complex structure in sounds. Our work builds on the spikegram representation of [5], thus avoiding the limitations arising from spectrogram-based methods, and makes a number of novel contributions. Unlike previous work [3, 4], the learned kernels are shiftable in both time *and* log-frequency, which enables the model to learn time- and frequency-relative patterns and use a small number of kernels efficiently to represent a wide variety of sound features. In addition, the model describes acoustic structure on multiple scales (via a hierarchical component and a recurrent component), which capture fundamentally different kinds of statistical regularities.

Technical contributions of ths work include methods for learning and performing approximate inference in a generalized linear model in which some of the inputs are unobserved and sparse (in this case the second-layer spikes). The computational framework developed here is general, and may have other applications in modeling sparse data with partially observed variables. Because the model is nonlinear, multi-layer cascades could lead to substantially more powerful models.

Applying the model to complex natural sounds (speech), we demonstrated that it can learn non-trivial features, and we have shown how these features can be composed to form basic acoustic units. We also showed a simple application to denoising, demonstrating improved performance to wavelet thresholding. The framework provides a general methodology for learning higher-order features of sounds, and we expect that it will prove useful in representing other structured sounds such as music, animal vocalizations, or ambient natural sounds.

### 6.1 Acknowledgments

We thank Richard Turner and Josh McDermott for helpful discussions.

# References

[1] C. Fevotte, B. Torresani, L. Daudet, and S. Godsill, "Sparse linear regression with structured priors and application to denoising of musical audio," *Audio, Speech, and Language Processing, IEEE Transactions on*, vol. 16, pp. 174 –185, jan. 2008.

[2] M. Plumbley, T. Blumensath, L. Daudet, R. Gribonval, and M. Davies, "Sparse representations in audio and music: From coding to source separation," *Proceedings of the IEEE*, vol. 98, pp. 995 –1005, june 2010.

[3] D. J. Klein, P. König, and K. P. Körding, "Sparse spectrotemporal coding of sounds," *EURASIP J. Appl. Signal Process.*, vol. 2003, pp. 659–667, Jan. 2003.

[4] H. Lee, Y. Largman, P. Pham, and A. Y. Ng, "Unsupervised feature learning for audio classification using convolutional deep belief networks," in *Advances in Neural Information Processing Systems*, pp. 1096–1104, The MIT Press, 2009.

[5] E. Smith and M. S. Lewicki, "Efficient coding of time-relative structure using spikes," *Neural Computation*, vol. 17, no. 1, pp. 19–45, 2005.

[6] M. Lewicki and T. Sejnowski, "Coding time-varying signals using sparse, shift-invariant representations," in *Advances in Neural Information Processing Systems*, pp. 730–736, The MIT Press, 1999.

[7] S. Mallat and Z. Zhang, "Matching pursuits with time-frequency dictionaries," *IEEE Trans Sig Proc*, vol. 41, pp. 3397–3415, December 1993.

[8] E. Smith and M. S. Lewicki, "Efficient auditory coding," *Nature*, vol. 439, no. 7079, 2006.

[9] P. McCullagh and J. A. Nelder, *Generalized linear models (Second edition)*. London: Chapman & Hall, 1989.

[10] J. S. Garofolo, L. F. Lamel, W. M. Fisher, J. G. Fiscus, D. S. Pallett, and N. L. Dahlgren, "Darpa timit acoustic phonetic continuous speech corpus cdrom," 1993.

[11] S. Mallat, *A Wavelet Tour of Signal Processing, Third Edition: The Sparse Way*. Academic Press, 3rd ed., 2008.

